# Learning Probabilistic Non-Linear Latent Variable Models for Tracking Complex Activities

**Angela Yao**[*]　　　Juergen Gall　　　Luc Van Gool　　　Raquel Urtasun
ETH Zurich　　　　ETH Zurich　　　　ETH Zurich　　　　TTI Chicago
{yaoa, gall, vangool}@vision.ee.ethz.ch, rurtasun@ttic.edu

## Abstract

A common approach for handling the complexity and inherent ambiguities of 3D human pose estimation is to use pose priors learned from training data. Existing approaches however, are either too simplistic (linear), too complex to learn, or can only learn latent spaces from "simple data", i.e., single activities such as walking or running. In this paper, we present an efficient stochastic gradient descent algorithm that is able to learn probabilistic non-linear latent spaces composed of multiple activities. Furthermore, we derive an incremental algorithm for the online setting which can update the latent space without extensive relearning. We demonstrate the effectiveness of our approach on the task of monocular and multi-view tracking and show that our approach outperforms the state-of-the-art.

## 1   Introduction

Tracking human 3D articulated motions from video sequences is well known to be a challenging machine vision problem. Estimating the human body's 3D location and orientation of the joints is notoriously difficult because it is a high-dimensional problem and is riddled with ambiguities coming from noise, monocular imagery and occlusions. To reduce the complexity of the task, it has become very popular to use prior models of human pose and dynamics [20, 25, 27, 28, 8, 13, 22].

Linear models (e.g. PCA) are among the simplest priors [20, 15, 26], though linearity also restricts a model's expressiveness and results in inaccuracies when learning complex motions. Priors generated from non-linear dimensionality reduction techniques such as Isomap [23] and LLE [18] have also been used for tracking [5, 8]. These techniques try to preserve the local structure of the manifold but tend to fail when manifold assumptions are violated, e.g., in the presence of noise, or multiple activities. Moreover, LLE and Isomap provide neither a probability distribution over the space of possible poses nor a mapping from the latent space to the high dimensional space. While such a distribution and or mapping can be learned *post hoc*, learning them separately from the latent space typically results in suboptimal solutions.

Probabilistic latent variable models (e.g. probabilistic PCA), have the advantage of taking uncertainties into account when learning latent representations. Taylor et al. [22] introduced the use of Conditional Restricted Boltzmann Machines (CRBM) and implicit mixtures of CRBM (imCRBM), which are composed of large collections of discrete latent variables. Unfortunately, learning this type of model is a highly complex task. A more commonly used latent variable model is the Gaussian Process Latent Variable Model (GPLVM) [9] which has been applied to animation [27] and tracking [26, 25, 6, 7]. While the GPLVM is very successful at modeling small training sets with single activities, it often struggles to learn latent spaces from larger datasets, especially those with multiple activities. The main reason is that the GPLVM is a non-parametric model; learning requires

---

[*]*This research was supported by the Swiss National Foundation NCCR project IM2, NSERC Canada and NSF #1017626.* Source code is available at www.vision.ee.ethz.ch/yaoa

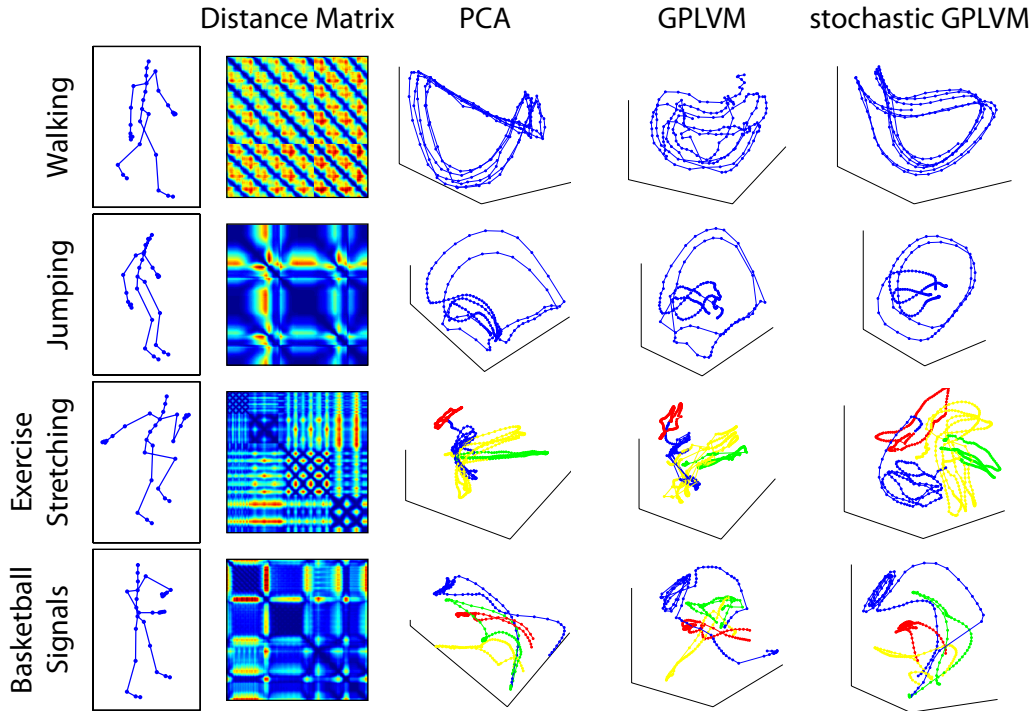

Figure 1: **Representative poses, data (Euclidean) distance matrices and learned latent spaces** from walking, jumping, exercise stretching and basketball signal sequences. GPLVM was initialized using probabilistic PCA; while stochastic GPLVM was initialized randomly.

the optimization of a non-convex function, for which complexity grows with the number of training samples. As such, having a good initialization is key for success [9], though good initializations are not always available [6], especially with complex data. Additionally, GPLVM learning scales cubicly with the number of training examples, and application to large datasets is computationally intractable, making it necessary to use sparsification techniques to approximate learning [17, 10]. As a consequence, the GPLVM has been mainly applied to single activities, e.g., walking or running.

More recent works have focused on handling multiple activities, most often with mixture models [14, 12, 13] or switching models [16, 8, 2]. However, coordinating the different components of the mixture models requires special care to ensure that they are aligned in the latent space [19], thereby complicating the learning process. In addition, both mixture and switching models require a discrete notion of activity which is not always available, e.g. dancing motions are not a discrete set. Others have tried to couple discriminate action classifiers with action-specific models [1, 5], though accuracy of such systems does not scale well with the number of actions.

A good prior model for tracking should be accurate, expressive enough to capture a wide range of human poses, and easy and tractable for both learning and inference. Unfortunately, none of the aforementioned approaches exhibit all of these properties. In this paper, we are interested in learning a probabilistic model that fulfill all of these criteria. Towards this end, we propose a stochastic gradient descent algorithm for the GPLVM which can learn latent spaces from random initializations. We draw inspiration for our work from two main sources. The first, [24], approximates Gaussian process regression for large training sets by doing online predictions based on local neighborhoods. The second, [11], maximizes the likelihood function for GPLVM by considering one dimension of the gradient at a time in the context of collaborative filtering. Based on these two works, we propose a similar strategy to approximate the gradient computation within each step of the stochastic gradient descent algorithm. Local estimation of the gradients allows our approach to efficiently learn models from large and complex training sets while mitigating the problem of local minima. Furthermore, we propose an online algorithm that can effectively learn latent spaces incrementally without extensive relearning. We demonstrate the effectiveness of our approach on the task of monocular and multi-view tracking and show that our approach outperforms the state-of-the-art on the standard benchmark HumanEva [21].

## 2  Stochastic learning

We first review the GPLVM, the basis of our work, and then introduce our optimization method for learning with stochastic local updates. Finally, we derive an extension of the algorithm which can be applied to the online setting.

### 2.1  GPLVM Review

The GPLVM assumes that the observed data has been generated by some unobserved latent random variables. More formally, let $\mathbf{Y} = [\mathbf{y}_1, \cdots, \mathbf{y}_N]^T$ be the set of observations $\mathbf{y}_i \in \Re^D$, and $\mathbf{X} = [\mathbf{x}_1, \cdots, \mathbf{x}_N]^T$ be the set of latent variables $\mathbf{x}_i \in \Re^Q$, with $Q \ll D$. The GPLVM relates the latent variables and the observations via the probabilistic mapping $y^{(d)} = f(\mathbf{x}) + \eta$, with $\eta$ being i.i.d. Gaussian noise, and $y^{(d)}$ the $d$-th coordinate of the observations. In particular, the GPLVM places a Gaussian process prior over the mapping $f$ such that marginalization of the mapping can be done in closed form. The resulting conditional distribution becomes

$$p\left(\mathbf{Y}|\mathbf{X}, \boldsymbol{\beta}\right) = \frac{1}{\sqrt{(2\pi)^{N \cdot D} |\mathbf{K}|^D}} \exp\left(-\frac{1}{2} tr\left(\mathbf{K}^{-1}\mathbf{Y}\mathbf{Y}^T\right)\right), \tag{1}$$

where $\mathbf{K}$ is the kernel matrix with elements $\mathbf{K}_{ij} = \mathbf{k}(\mathbf{x}_i, \mathbf{x}_j)$ and the kernel $\mathbf{k}$ has parameters $\boldsymbol{\beta}$. Here, we follow existing approaches [26, 25] and use a a kernel compounded from an RBF, a bias, and Gaussian noise, i.e., $\mathbf{k}(\mathbf{x}, \mathbf{x}') = \beta_1 \exp\left(-\frac{\|\mathbf{x}-\mathbf{x}'\|^2}{\beta_2}\right) + \beta_3 + \frac{\delta_{x,x'}}{\beta_4}$.

The GPLVM is usually learned by maximum likelihood estimation of the latent coordinates $\mathbf{X}$ and the kernel hyperparameters $\boldsymbol{\beta} = \{\beta_1, \cdots, \beta_4\}$. This is equivalent to minimizing the negative log likelihood $\mathcal{L}$:

$$\mathcal{L} = -\ln p\left(\mathbf{Y}|\mathbf{X}, \boldsymbol{\beta}\right) = -\frac{DN}{2}\ln 2\pi - \frac{D}{2}\ln|\mathbf{K}| - \frac{1}{2}tr\left(\mathbf{K}^{-1}\mathbf{Y}\mathbf{Y}^T\right). \tag{2}$$

Typically a gradient descent algorithm is used for the minimization. The gradient of $\mathcal{L}$ with respect to $\mathbf{X}$ can be obtained via the chain rule, where

$$\frac{\partial \mathcal{L}}{\partial \mathbf{X}} = \frac{\partial \mathcal{L}}{\partial \mathbf{K}} \cdot \frac{\partial \mathbf{K}}{\partial \mathbf{X}} = -\left(\mathbf{K}^{-1}\mathbf{Y}\mathbf{Y}^T\mathbf{K}^{-1} - D\mathbf{K}^{-1}\right) \cdot \frac{\partial \mathbf{K}}{\partial \mathbf{X}}. \tag{3}$$

Similarly, the gradient of $\mathcal{L}$ with respect to $\boldsymbol{\beta}$ can be found by substituting $\frac{\partial K}{\partial \mathbf{X}}$ with $\frac{\partial K}{\partial \boldsymbol{\beta}}$ in Eq. (3) (see [9] for the exact derivation). As $N$ gets large, however, computing the gradients becomes computationally expensive, because inverting $\mathbf{K}$ is of $O(N^3)$, with $N$ the number of training examples. More importantly, as the negative log likelihood $\mathcal{L}$ is highly non-convex, especially with respect to $\mathbf{X}$, standard gradient descent approaches tend to get stuck in local minima, and rely on having good initializations for success.

We now demonstrate how a stochastic gradient descent approach can be used to reduce computational complexity as well as decrease the chances of getting trapped in local minima. In particular, as shown in our experiments (Section 3), we are able to obtain smooth and accurate manifolds (see Fig. 1) from random initialization.

### 2.2  Stochastic Gradient Descent

In standard gradient descent, all points are taken into account at the same time when computing the gradient; stochastic gradient descent approaches, on the other hand, approximate the gradient at each point individually. Typically, a loop goes over the points in a series or by randomly sampling from the training set. Note that after iterating over all the points, the gradient is exact. As the GPLVM is a non-parametric approach, the gradient computation at each point does not decompose, making it necessary to invert $\mathbf{K}$, an $O(N^3)$ operation at every iteration. We propose, however, to approximate the gradient computation within each step of the stochastic gradient descent algorithm. Therefore, the gradient of $\mathcal{L}$ can be estimated locally for some neighborhood of points $\mathbf{X}_R$, centered at a reference point $\mathbf{x}_r$, rather than over all of $\mathbf{X}$. Eq. (3) can then be evaluated only for the points within the neighborhood, i.e.,

| **Algorithm 1: Stochastic GPLVM** |
|---|
| Randomly initialize $\mathbf{X}$ |
| Set $\boldsymbol{\beta}$ with an initial guess |
| **for** $t$ **= 1:T** |
| $\quad$ randomly select $\mathbf{x}_r$ |
| $\quad$ find $R$ neighbors around $\mathbf{x}_r$: $\mathbf{X}_R = \mathbf{X} \in \mathcal{R}$ |
| $\quad$ Compute $\frac{\partial L}{\partial \mathbf{X}_R}$ and $\frac{\partial L}{\partial \boldsymbol{\beta}_R}$ (see Eq. (3)) |
| $\quad$ Update $\mathbf{X}$ and $\boldsymbol{\beta}$: |
| $\quad\quad \Delta \mathbf{X}_t = \mu_X \cdot \Delta \mathbf{X}_{t-1} + \eta_X \cdot \frac{\partial L}{\partial \mathbf{X}_R}$ |
| $\quad\quad \mathbf{X}_t \leftarrow \mathbf{X}_{t-1} + \Delta \mathbf{X}_t$ |
| $\quad\quad \Delta \boldsymbol{\beta}_t = \mu_\beta \cdot \Delta \boldsymbol{\beta}_{t-1} + \eta_\beta \cdot \frac{\partial L}{\partial \boldsymbol{\beta}_R}$ |
| $\quad\quad \boldsymbol{\beta}_t \leftarrow \boldsymbol{\beta}_{t-1} + \Delta \boldsymbol{\beta}_t$ |
| **end** |

| **Algorithm 2: Incremental stochastic GPLVM** |
|---|
| **for** $t = 1 : T_1$ |
| $\quad$ Learn $\mathbf{X}_{orig}$ and $\boldsymbol{\beta}_{orig}$ as per Algorithm 1. |
| **end** |
| Initialize $\mathbf{X}_{incr}$ using nearest neighbors. |
| Set $\boldsymbol{\beta} = \boldsymbol{\beta}_{orig}$ |
| Group data: |
| $\quad \mathbf{Y} = [\mathbf{Y}_{orig}, \mathbf{Y}_{incr}]$ |
| $\quad \mathbf{X} = [\mathbf{X}_{orig}, \mathbf{X}_{incr}]$ |
| **for** $t = T_1 + 1 : T_2$ |
| $\quad$ randomly select $\mathbf{x}_r \in \mathbf{X}_{incr}$ |
| $\quad$ find $R$ neighbors around $\mathbf{x}_r$: $\mathbf{X}_R = \mathbf{X} \in \mathcal{R}$ |
| $\quad$ Compute $\frac{\partial L_{incr}}{\partial \mathbf{X}_R}$ and $\frac{\partial L_{incr}}{\partial \boldsymbol{\beta}_R}$ (see Eq. (6)) |
| $\quad$ Update $\mathbf{X}$ and $\boldsymbol{\beta}$: |
| $\quad\quad \Delta \mathbf{X}_t = \mu_X \cdot \Delta \mathbf{X}_{t-1} + \eta_X \cdot \frac{\partial L_{incr}}{\partial \mathbf{X}_R}$ |
| $\quad\quad \mathbf{X}_t \leftarrow \mathbf{X}_{t-1} + \Delta \mathbf{X}_t$ |
| $\quad\quad \Delta \boldsymbol{\beta}_t = \mu_\beta \cdot \Delta \boldsymbol{\beta}_{t-1} + \eta_\beta \cdot \frac{\partial L_{incr}}{\partial \boldsymbol{\beta}_R}$ |
| $\quad\quad \boldsymbol{\beta}_t \leftarrow \boldsymbol{\beta}_{t-1} + \Delta \boldsymbol{\beta}_t$ |
| **end** |

Figure 2: **Stochastic gradient descent and incremental learning** for the GPLVM; $\mu_{(\cdot)}$ is a momentum parameter and $\eta_{(\cdot)}$ is the learning rate. Note that $R$, $\mu$, and $\eta$ can also vary with $t$.

$$\frac{\partial \mathcal{L}}{\partial \mathbf{X}_R} \approx - \left( \mathbf{K}_R^{-1} \mathbf{Y}_R \mathbf{Y}_R^T \mathbf{K}_R^{-1} - D \mathbf{K}_R^{-1} \right) \cdot \frac{\partial \mathbf{K}_R}{\partial \mathbf{X}_R}, \tag{4}$$

where $\mathbf{K}_R$ is the kernel matrix for $\mathbf{X}_R$ and $\mathbf{Y}_R$ is the corresponding neighborhood data points.

We employ a random strategy for choosing the reference point $\mathbf{x}_r$. The neighborhood $\mathcal{R}$ can be determined by any type of distance measure, such as Euclidean distance in the latent space and/or data space, or temporal neighbors when working with time series. More critical than the specific type of distance measure, however, is allowing sufficient coverage of the latent space so that each neighborhood is not restricted too locally. To keep the complexity low, it is beneficial to sample randomly from a larger set of neighbors (see supplementary material).

The use of stochastic gradient descent has several desirable traits that correct for the aforementioned drawbacks of GPLVMs. First, computational complexity is greatly reduced, making it feasible to learn latent spaces with much larger amounts of data. Secondly, estimating the gradients stochastically and locally improves robustness of the learning process against local minima, making it possible to have a random initialization. An algorithmic summary of stochastic gradient descent learning for GPLVMs is given in Fig. 2.

## 2.3 Incremental Learning

In this section, we derive an incremental learning algorithm based on the stochastic gradient descent approach of the previous section. In this setting, we have an initial model which we would like to update as new data comes in on the fly. More formally, let $\mathbf{Y}_{orig}$ be the initial training data, and $\mathbf{X}_{orig}$ and $\boldsymbol{\beta}_{orig}$ be a model learned from $\mathbf{Y}_{orig}$ using stochastic GPLVM. For every step in the online learning, let $\mathbf{Y}_{incr}$ be new data, which can be as little as a single point or an entire set of training points. Let $\mathbf{Y} = [\mathbf{Y}_{orig}, \mathbf{Y}_{incr}] \in \mathbb{R}^{(N+M) \times D}$ be the set of training points containing both the already trained data $\mathbf{Y}_{orig}$, and the new incoming data $\mathbf{Y}_{incr}$, and let $\mathbf{X} = [\mathbf{X}_{orig}, \mathbf{X}_{incr}] \in \mathbb{R}^{(N+M) \times Q}$ be the corresponding latent coordinates, where $M$ is the number of newly added training examples. Let $\hat{\mathbf{X}}_{orig}$ be the estimate of the latent coordinates that has already been learned.

A possible strategy is to update only the incoming points; however, we would like to exploit the new data for improving the estimate of the entire manifold, therefore we propose to learn the full $\mathbf{X}$. To prevent the already-learned manifold from diverging and also to speed up learning, we add a regularizer to the log-likelihood to encourage original points to not deviate too far from their initial estimate. To this end, we use the Frobenius norm of the deviation from the estimate $\hat{\mathbf{X}}_{orig}$. Learning

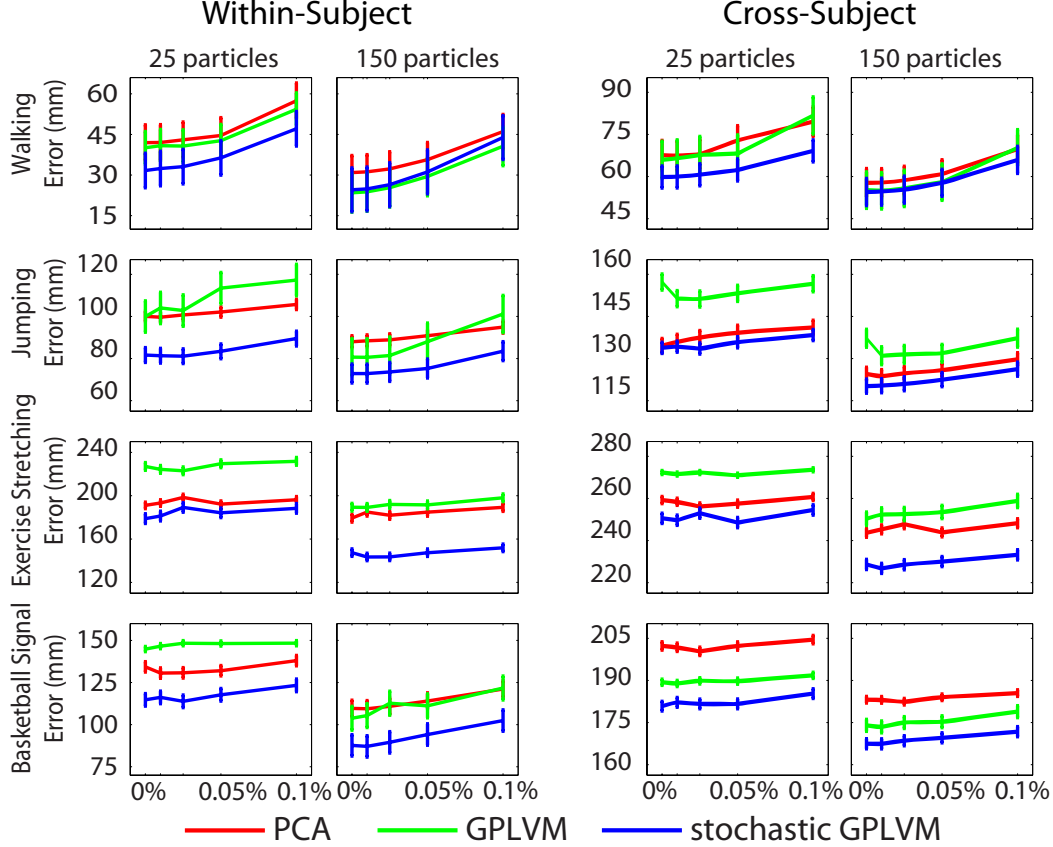

Figure 3: **Within- and cross-subject 3D tracking errors** for each type of activity sequence with respect to amount of additive noise for different number of particles, where error bars represent the standard deviation from repetitions runs.

is then done by minimizing the regularized negative log-likelihood

$$\mathcal{L}_{incr} = \mathcal{L} + \lambda \cdot \frac{1}{N} ||\mathbf{X}_{1:N,:} - \hat{\mathbf{X}}_{orig}||^2_F \ . \tag{5}$$

Here, $\mathbf{X}_{1:N,:}$ indicates the first $N$ rows of $\mathbf{X}$, while $\lambda$ is a weighting on the regularization term. The gradient of $\mathcal{L}$ with respect to $\mathbf{X}_R$ [1] can then be computed as

$$\frac{\partial \mathcal{L}_{incr}}{\partial \mathbf{X}_R} = \frac{\partial \mathcal{L}}{\partial \mathbf{X}_R} + \lambda \cdot \frac{2}{N} \cdot \left( \mathbf{X}_{1:N,:} - \hat{\mathbf{X}}_{orig} \right) \frac{\partial \mathbf{X}_{1:N,:}}{\partial \mathbf{X}_R}. \tag{6}$$

We employ a stochastic gradient descent approach for our incremental learning, where the points are sampled randomly from $\mathbf{X}_{incr}$. Note that while $\mathbf{x}_r$ is only sampled from $\mathbf{X}_{incr}$ in the subsequent learning step, this does not exclude points in $\mathbf{X}_{orig}$ from being a part of the neighbourhood $\mathcal{R}$, and thus from being updated. We have chosen a nearest neighbor approach by comparing $\mathbf{Y}_{incr}$ to $\mathbf{Y}_{orig}$ for estimating an initial $\mathbf{X}_{incr}$, though other possibilities include performing a grid search in the latent space and selecting locations with the highest global log-likelihood (Eq. (2)) or training a regressor from $\mathbf{Y}_{orig}$ to $\mathbf{X}_{orig}$ to be applied to $\mathbf{Y}_{incr}$. An algorithmic summary of the incremental method is provided in Fig. 2.

## 2.4 Tracking Framework

During training, a latent variable model $\mathcal{M}$ is learned from $\mathbf{Y}_M$, where $\mathbf{Y}_M$ are relative joint locations with respect to a root node. We designate the learned latent points as $\mathbf{X}_M$. During inference, tracking is performed in the latent space using a particle filter. The corresponding pose is computed by projecting back to the data space via the Gaussian process mapping learned in the GPLVM.

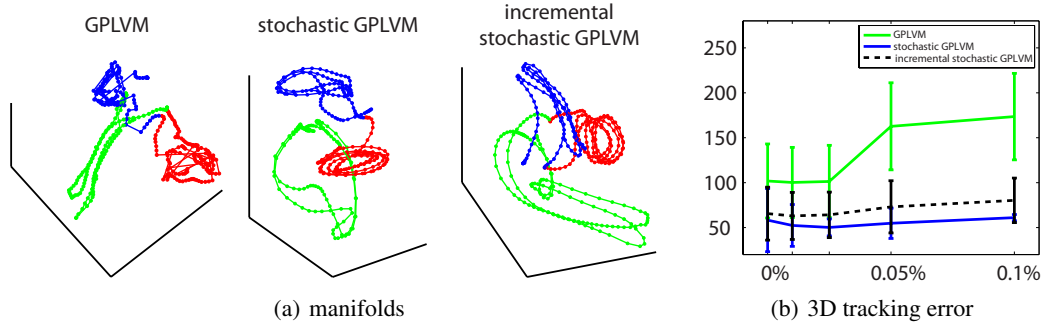

<center>(a) manifolds                               (b) 3D tracking error</center>

Figure 4: *(a)* Learned manifolds from **regular GPLVM, stochastic GPLVM and incremental stochastic GPLVM** from an exercise stretching sequence, where blue, red, green indicate jumping jacks, jogging and squats respectively and *(b)* the associated 3D tracking errors (mm), where error bars indicate standard deviation over repeated runs.

We model the state $\mathbf{s}$ at time $t$ as $\mathbf{s}_t = (\mathbf{x}_t, \mathbf{g}_t, \mathbf{r}_t)$ where $\mathbf{x}_t$ denotes position in the latent space, while $\mathbf{g}_t$ and $\mathbf{r}_t$ are the global position and rotation of the root node. Particles are initialized in the latent space by a nearest neighbor search between the observed 2D image pose in the first frame of the sequence and the projected 2D poses of $\mathbf{Y}_M$. Particles are then propagated from frame to frame using a first-order Markov model

$$\mathbf{x}_t^i = \mathbf{x}_{t-1}^i + \dot{\mathbf{x}}_t^i, \qquad \mathbf{g}_t^i = \mathbf{g}_{t-1}^i + \dot{\mathbf{g}}_t^i, \qquad \mathbf{r}_t^i = \mathbf{r}_{t-1}^i + \dot{\mathbf{r}}_t^i. \tag{7}$$

We approximate the derivative $\dot{\mathbf{x}}^i$ with the difference between temporally sequential points of the nearest neighbors in $\mathbf{X}_M$, while $\dot{\mathbf{g}}^i$ and $\dot{\mathbf{r}}^i$ are drawn from individual Gaussians with means and standard deviations estimated from the training data. The tracked latent position $\hat{\mathbf{x}}_t$ at time $t$ is then approximated as the mode over all particles in the latent space while $\hat{\mathbf{y}}_t$ is estimated via the mean Gaussian process estimate

$$\hat{\mathbf{y}}_t = \mu_M + \mathbf{Y}_M^T \mathbf{K}^{-1} \mathbf{k}(\hat{\mathbf{x}}_t, \mathbf{X}_M), \tag{8}$$

with $\mu_M$ the mean of $\mathbf{Y}_M$ and $\mathbf{k}(\hat{\mathbf{x}}_t, \mathbf{X}_M)$ the vector with elements $\mathbf{k}(\hat{\mathbf{x}}_t, \mathbf{x}_m)$ for all $\mathbf{x}_m$ in $\mathbf{X}_M$. Note that the computation of $\mathbf{K}^{-1}$ needs to be performed only once and can be stored.

## 3 Experimental Evaluation

We demonstrate the effectiveness of our model when applied to tracking in both monocular and multi-view scenarios. In all cases, the latent models were learned with $\mu_X = 0.8$, $\mu_\beta = 0.5$, $\eta_X$=10e-4, $\eta_\beta$ =10e-8; we annealed these parameters over the iterations. To further smooth the learned models, we incorporate a Gaussian Process prior over the dynamics of the training data in the latent space [27] for the GPLVM and the stochastic GPLVM. *We refer the reader to the supplementary material for a visualization of the learning process as well as the results.*

### 3.1 Monocular Tracking

We compare in the monocular setting the use of PCA, regular GPLVM and our stochastic GPLVM to learn latent spaces from motion capture sequences (from the CMU Motion Capture Database [3]). We chose simple single-activity sequences, such as walking (3 subjects, 18 sequences) and jumping (2 subjects, 8 sequences), as well as complex multi-activity sequences, such as stretching exercises (2 subjects, 6 sequences) and basketball refereeing signals (7 subjects, 13 sequences). The stretching exercise and basketball signal sequences were cut to each contain four types of activities. We synthesized 2D data by projecting the mocap from 3D to 2D and then corrupting the location of each joint with different levels of additive Gaussian noise. We then recover the 3D locations of each joint from the noisy images by tracking with the particle filter described in the previous section.

Examples of learned latent spaces for each type of sequence (i.e., walking, jumping, exercise, basketball) are shown in Fig. 1. We used a neighborhood of 60 points for the single activity sequences, which have on average 250 training examples, and 100 points for the multiple activity sequences,

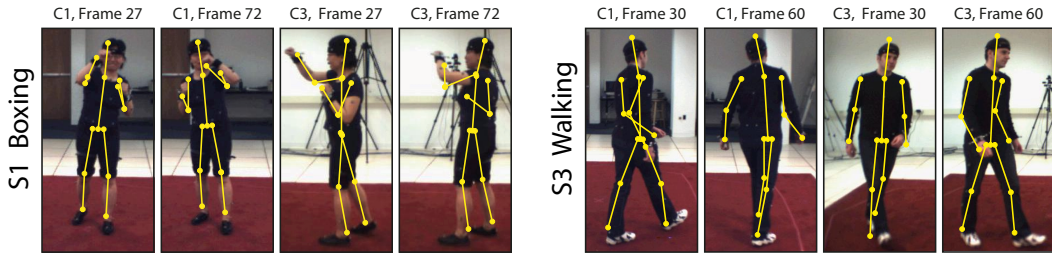

Figure 5: Example poses from **tracked results on HumanEva**.

which have on average 800 training examples. For a sequence of 800 training examples, the stochastic GPLVM takes only 27s to learn (neighborhood of 100 points, 2500 iterations); in comparison, the regular GPLVM takes 2560s for 312 iterations, while with FITC approximations [10] takes on average 1700s (100 active points, 2500 iterations)[2]. In general, as illustrated by Fig. 1, the manifolds learned with stochastic GPLVM have smoother trajectories than those learned from PCA and GPLVM, with better separation between the activities in the multi-activity sequences.

We evaluate the effectiveness of the learned latent pose models for tracking by comparing the average tracking error per joint per frame between PCA, GPLVM and stochastic GPLVM in two sets of experiments. In the first, training and test sequences are performed by the same subject; in the second, to test generalization properties of the different latent spaces, we train and test on different subjects. We report results average over 10 sequences, each repeated over 10 different runs of the tracker. We use importance sampling and weight each particle at time $t$ proportionally to a likelihood defined by the reprojection error: $w_t^i \propto \exp\left(-\alpha \sum_j \|p_{j,t}^i - q_{j,t}\|^2\right)$, where $p_{j,t}^i$ is the projected 2D position of joint $j$ in $\mathbf{y}_t^i$ from $\mathbf{x}_t^i$ (see Eq. (8)) and $q_{j,t}$ is the observed 2D position of joint $j$, assuming that the camera projection and correspondences between joints are already known. $\alpha$ is a parameter determining selectivity of the weight function (we use $\alpha = 5 \cdot 10^{-5}$).

Fig. 3 depicts 3D tracking error as a function of the amount of Gaussian noise for different number of particles employed in the particle filter for the within- and cross-subject experiments. As expected, tracking error is lower within-subject than cross-subject for all types of latent models. For the simple activities such as walking and jumping, GPLVM generally outperforms PCA, but for the complex activities, it performs only comparably or worse than PCA (with the exception of cross-subject basketball signals). Our stochastic GPLVM, on the other hand, consistently outperforms PCA and matches or outperforms the regular GPLVM in all experimental conditions, with significantly better performance in the complex, multi-activity sequences. Additional experiments are provided in the supplementary material.

### 3.2 Online Tracking

We took two stretching exercise sequences with three different activities from the same subject and apply the online learning algorithm (see Sec. 2.3), setting $\lambda = 2$. We consider each activity as a new batch of data, and learn the latent space on the first sequence and then track on the second and vice versa. We find the online algorithm less accurate for tracking than the stochastic GPLVM learned with all data. This is expected since the latent space is biased towards the initial set of activities. We note, however, that the incremental stochastic GPLVM still outperforms the regular GPLVM, as illustrated in Fig. 4(b). Examples of the learned manifolds are shown in Fig. 4(a).

### 3.3 Multi-view Tracking on HumanEva

We also evaluate our learning algorithm on the HumanEva benchmark [21] on the activities walking and boxing. For all experiments, we use a particle filter as described in Sec. 2.4 with 25 particles as well as an additional annealing component [4] of 15 layers. To maintain consistency with previous

| Train | Test | [28] | [13] | GPLVM | CRBM [22] | imCRBM [22] | Ours |
|---|---|---|---|---|---|---|---|
| S1 | S1 | - | - | $57.6 \pm 11.6$ | $48.8 \pm 3.7$ | $58.6 \pm 3.9$ | $\mathbf{44.0 \pm 1.8}$ |
| S1,2,3 | S1 | 140.3 | - | $64.3 \pm 19.2$ | $55.4 \pm 0.8$ | $54.3 \pm 0.5$ | $\mathbf{41.6 \pm 0.8}$ |
| S2 | S2 | - | $68.7 \pm 24.7$ | $98.2 \pm 15.8$ | $\mathbf{47.4 \pm 2.9}$ | $67.0 \pm 0.7$ | $54.4 \pm 1.8$ |
| S1,2,3 | S2 | 149.4 | - | $155.9 \pm 48.8$ | $99.1 \pm 23.0$ | $69.3 \pm 3.3$ | $\mathbf{64.0 \pm 2.9}$ |
| S3 | S3 | - | $69.6 \pm 22.2$ | $71.6 \pm 10.0$ | $49.8 \pm 2.2$ | $51.4 \pm 0.9$ | $\mathbf{45.4 \pm 1.1}$ |
| S1,2,3 | S3 | 156.3 | - | $123.8. \pm 16.7$ | $70.9 \pm 2.1$ | $\mathbf{43.4 \pm 4.1}$ | $46.5 \pm 1.4$ |

Table 1: Comparison of 3D tracking errors (mm) on the entire walking validation sequence with subject-specific models, where $\pm$ indicates standard deviation over runs, except for [13], who reports tracking results for 200 frames of the sequences, with standard deviation over frames.

| Model | Tracking Error |
|---|---|
| [16] as reported in [12] | $569.90 \pm 209.18$ |
| [14] as reported in [12] | $380.02 \pm 74.97$ |
| GPLVM | $121.44 \pm 30.7$ |
| [12] | $117.0 \pm 5.5$ |
| Best CRBM [22] | $75.4 \pm 9.7$ |
| Ours | $\mathbf{74.1 \pm 3.3}$ |

Table 2: Comparison of 3D tracking errors (mm) on boxing validation sequence for S1, where $\pm$ indicates standard deviation over runs. Our results are comparable to the state-of-the-art [22].

works, we use the images from the 3 color cameras and the simple silhouette and edge likelihoods provided in the HumanEva baseline algorithm [21].

*HumanEva-I Walking:* As per [22, 28, 13], we track the walking validation sequences of subjects S1, S2, and S3. The latent variable models are learned on the training sequences, being either subject-specific or with all three subjects combined. Subject-specific models have $\sim$1200-2000 training examples each, for which we used a neighborhood of 60 points, while the combined model has $\sim$4000 training examples with a neighborhood of 150 points. 3D tracking errors averaged over the 15 joints as specified in [21] and over all frames in the full sequence are depicted in Table1. Sample frames of the estimated poses are shown in Fig. 5. In four of the six training/test combinations, the stochastic GPLVM model outperforms the state-of-the-art CRBM and imCRBM model from [22], while in the other two cases, our model is comparable. These results are remarkable, given that we use only a simple first-order Markov model for estimating dynamics, and our success can only be attributed to the latent model's accuracy in encoding the body poses from the training data.

*HumanEva-I boxing:* We also track the validation sequence of S1 for boxing to assess the ability of the stochastic GPLVM for learning acyclic motions. 3D tracking errors are shown in Table 2 and are compared with [14, 13, 22]. Our results are slightly better than state-of-the-art.

## 4   Conclusion and Future Work

In this paper, we try to learn a probabilistic prior model which is accurate yet expressive, and is tractable for both learning and inference. Our proposed stochastic GPLVM fulfills all these criteria - it effectively learns latent spaces of complex multi-activity datasets in a computationally efficient manner. When applied to tracking, our model outperforms state-of-the-art on the HumanEva benchmark, despite the use of very few particles and only a simple first-order Markov model for handling dynamics. In addition, we have also derived a novel approach for learning latent spaces incrementally. One of the great criticisms of current latent variable models is that they cannot handle new training examples without relearning; given the sometimes cumbersome learning process, this is not always feasible. Our incremental method can be easily applied to an online setting without extensive relearning, which may have impact in applications such as robotics where domain adaptation might be key for accurate prediction. In the future, we plan to further investigate the incorporation of dynamics into the stochastic model, particularly for multiple activities.

## Footnotes

[1] $\frac{\partial \mathcal{L}_{incr}}{\partial \boldsymbol{\beta}_R} = \frac{\partial \mathcal{L}}{\partial \boldsymbol{\beta}_R}$ since the regularization term does not depend on $\boldsymbol{\beta}_R$.

[2]Note that none of the models have completed training. For timing purposes, we take here a fixed number of iterations for the stochastic method and the FITC approximation and the "equivalent" for the regular GPLVM, i.e., 2500 iterations /8, where 8 comes from the fact that $8X$ more points are used in computing $\mathbf{K}$.

# References

[1] A. Baak, M. Mueller B. Rosenhahn, and H.-P. Seidel. Stabilizing motion tracking using retrieved motion priors. In *ICCV*, 2009.

[2] J. Chen, M. Kim, Y. Wang, and Q. Ji. Switching gaussian process dynamic models for simultaneous composite motion tracking and recognition. In *CVPR*, 2009.

[3] CMU Mocap Database. http://mocap.cs.cmu.edu/.

[4] J. Deutscher and I. Reid. Articulated body motion capture by stochastic search. *IJCV*, 61(2), 2005.

[5] J. Gall, A. Yao, and L. Van Gool. 2d action recognition serves 3d human pose estimation. In *ECCV*, 2010.

[6] A. Geiger, R. Urtasun, and T. Darrell. Rank priors for continuous non-linear dimensionality reduction. In *CVPR*, 2009.

[7] S. Hou, A. Galata, F. Caillette, N. Thacker, and P. Bromiley. Real-time body tracking using a gaussian process latent variable model. *ICCV*, 2007.

[8] T. Jaeggli, E. Koller-Meier, and L. Van Gool. Learning generative models for multi-activity body pose estimation. *IJCV*, 83(2):121–134, 2009.

[9] N. Lawrence. Probabilistic non-linear principal component analysis with gaussian process latent variable models. *JMLR*, 6:1783–1816, 2005.

[10] N. Lawrence. Learning for larger datasets with the gaussian process latent variable model. In *AISTATS*, 2007.

[11] N. Lawrence and R. Urtasun. Non-linear matrix factorization with gaussian processes. In *ICML*, 2009.

[12] R. Li, T. Tian, and S. Sclaroff. Simultaneous learning of non-linear manifold and dynamical models for high-dimensional time series. In *ICCV*, 2007.

[13] R. Li, T.-P. Tian, S. Sclaroff, and M.-H. Yang. 3d human motion tracking with a coordinated mixture of factor analyzers. *IJCV*, 87:170–190, 2010.

[14] R.S. Lin, C.B. Liu, M.H. Yang, N. Ahja, and S. Levinson. Learning nonlinear manifolds from time series. In *ECCV*, 2006.

[15] D. Ormoneit, C. Lemieux, and D. Fleet. Lattice particle filters. In *UAI*, 2001.

[16] V. Pavlovic, J. Rehg, and J. Maccormick. Learning switching linear models of human motion. In *NIPS*, pages 981–987, 2000.

[17] J. Quinonero-Candela and C. Rasmussen. A unifying view of sparse approximate gaussian process regression. *JMLR*, page 2005, 2006.

[18] S. Roweis and L. Saul. Nonlinear Dimensionality Reduction by Locally Linear Embedding. *Science*, 290(5500):2323–2326, 2000.

[19] S. Roweis, L. Saul, and G. Hinton. Global coordination of local linear models. In *NIPS*, 2002.

[20] H. Sidenbladh, M. Black, and D. Fleet. Stochastic tracking of 3d human figures using 2d image motion. In *ECCV*, 2000.

[21] L. Sigal, A. Balan, and M. Black. Humaneva: Synchronized video and motion capture dataset and baseline algorithm for evaluation of articulated human motion. *IJCV*, 87(1-2):4–27, 2010.

[22] G. Taylor, L. Sigal, D. Fleet, and G. Hinton. Dynamical binary latent variable models for 3d human pose tracking supplementary material. In *CVPR*, 2010.

[23] J. Tenenbaum, V. de Silva, and J. Langford. A Global Geometric Framework for Nonlinear Dimensionality Reduction. *Science*, 2000.

[24] R. Urtasun and T. Darrell. Sparse probabilistic regression for activity-independent human pose inference. In *CVPR*, 2008.

[25] R. Urtasun, D. Fleet, and P. Fua. 3d people tracking with gaussian process dynamical models. In *CVPR*, 2006.

[26] R. Urtasun, D. Fleet, A. Hertzman, and P. Fua. Priors for people tracking from small training sets. In *ICCV*, 2005.

[27] J. Wang, D. Fleet, and A. Hertzmann. Gaussian process dynamical models for human motion. *PAMI*, 30(2):283–298, 2008.

[28] X. Xu and B. Li. Learning motion correlation for tracking articulated human body with a rao-blackwellised particle filter. In *ICCV*, 2007.

